# Progressive mixture rules are deviation suboptimal

**Jean-Yves Audibert**
Willow Project - Certis Lab
ParisTech, Ecole des Ponts
77455 Marne-la-Vallée, France
audibert@certis.enpc.fr

## Abstract

We consider the learning task consisting in predicting as well as the best function in a finite reference set $\mathcal{G}$ up to the smallest possible additive term. If $R(g)$ denotes the generalization error of a prediction function $g$, under reasonable assumptions on the loss function (typically satisfied by the least square loss when the output is bounded), it is known that the progressive mixture rule $\hat{g}$ satisfies

$$\mathbb{E}R(\hat{g}) \leq \min_{g \in \mathcal{G}} R(g) + \mathrm{Cst}\, \frac{\log |\mathcal{G}|}{n}, \qquad (1)$$

where $n$ denotes the size of the training set, and $\mathbb{E}$ denotes the expectation w.r.t. the training set distribution. This work shows that, surprisingly, for appropriate reference sets $\mathcal{G}$, the deviation convergence rate of the progressive mixture rule is no better than $\mathrm{Cst}\,/\sqrt{n}$: it fails to achieve the expected $\mathrm{Cst}\,/n$. We also provide an algorithm which does not suffer from this drawback, and which is optimal in both deviation and expectation convergence rates.

## 1 Introduction

*Why are we concerned by deviations?* The efficiency of an algorithm can be summarized by its expected risk, but this does not precise the fluctuations of its risk. In several application fields of learning algorithms, these fluctuations play a key role: in finance for instance, the bigger the losses can be, the more money the bank needs to freeze in order to alleviate these possible losses. In this case, a "good" algorithm is an algorithm having not only low expected risk but also small deviations.

*Why are we interested in the learning task of doing as well as the best prediction function of a given finite set?* First, one way of doing model selection among a finite family of submodels is to cut the training set into two parts, use the first part to learn the best prediction function of each submodel and use the second part to learn a prediction function which performs as well as the best of the prediction functions learned on the first part of the training set. This scheme is very powerful since it leads to theoretical results, which, in most situations, would be very hard to prove without it. Our work here is related to the second step of this scheme.

Secondly, assume we want to predict the value of a continuous variable, and that we have many candidates for explaining it. An input point can then be seen as the vector containing the prediction of each candidate. The problem is what to do when the dimensionality $d$ of the input data (equivalently the number of prediction functions) is much higher than the number of training points $n$. In this setting, one cannot use linear regression and its variants in order to predict as well as the best candidate up to a small additive term. Besides, (penalized) empirical risk minimization is doomed to be suboptimal (see the second part of Theorem 2 and also [1]).

As far as the expected risk is concerned, the only known correct way of predicting as well as the best prediction function is to use the progressive mixture rule or its variants. These algorithms are introduced in Section 2 and their main good property is given in Theorem 1. In this work we prove that they do not work well as far as risk deviations are concerned (see the second part of Theorem

3). We also provide a new algorithm for this 'predict as well as the best' problem (see the end of Section 4).

## 2   The progressive mixture rule and its variants

We assume that we observe $n$ pairs of input-output denoted $Z_1 = (X_1, Y_1), \ldots, Z_n = (X_n, Y_n)$ and that each pair has been independently drawn from the same unknown distribution denoted $P$. The input and output spaces are denoted respectively $\mathcal{X}$ and $\mathcal{Y}$, so that $P$ is a probability distribution on the product space $\mathcal{Z} \triangleq \mathcal{X} \times \mathcal{Y}$. The quality of a (prediction) function $g : \mathcal{X} \to \mathcal{Y}$ is measured by the *risk* (or generalization error):

$$R(g) = \mathbb{E}_{(X,Y) \sim P} \, \ell[Y, g(X)],$$

where $\ell[Y, g(X)]$ denotes the loss (possibly infinite) incurred by predicting $g(X)$ when the true output is $Y$. We work under the following assumptions for the data space and the loss function $\ell : \mathcal{Y} \times \mathcal{Y} \to \mathbb{R} \cup \{+\infty\}$.

**Main assumptions.**   The input space is assumed to be infinite: $|\mathcal{X}| = +\infty$. The output space is a non-trivial (i.e. infinite) interval of $\mathbb{R}$ symmetrical w.r.t. some $a \in \mathbb{R}$: for any $y \in \mathcal{Y}$, we have $2a - y \in \mathcal{Y}$. The loss function is

- *uniformly exp-concave:* there exists $\lambda > 0$ such that for any $y \in \mathcal{Y}$, the set $\big\{y' \in \mathbb{R} : \ell(y, y') < +\infty\big\}$ is an interval containing $a$ on which the function $y' \mapsto e^{-\lambda \ell(y, y')}$ is concave.
- *symmetrical:* for any $y_1, y_2 \in \mathcal{Y}$, $\ell(y_1, y_2) = \ell(2a - y_1, 2a - y_2)$,
- *admissible:* for any $y, y' \in \mathcal{Y} \cap \, ]a; +\infty[$, $\ell(y, 2a - y') > \ell(y, y')$,
- *well behaved at center:* for any $y \in \mathcal{Y} \cap \, ]a; +\infty[$, the function $\ell_y : y' \mapsto \ell(y, y')$ is twice continuously differentiable on a neighborhood of $a$ and $\ell'_y(a) < 0$.

These assumptions imply that

- $\mathcal{Y}$ has necessarily one of the following form: $]-\infty; +\infty[$, $[a - \zeta; a + \zeta]$ or $]a - \zeta; a + \zeta[$ for some $\zeta > 0$.
- for any $y \in \mathcal{Y}$, from the exp-concavity assumption, the function $\ell_y : y' \mapsto \ell(y, y')$ is convex on the interval on which it is finite[1]. As a consequence, the risk $R$ is also a convex function (on the convex set of prediction functions for which it is finite).

The assumptions were motivated by the fact that they are satisfied in the following settings:

- least square loss with bounded outputs: $\mathcal{Y} = [y_{\min}; y_{\max}]$ and $\ell(y_1, y_2) = (y_1 - y_2)^2$. Then we have $a = (y_{\min} + y_{\max})/2$ and may take $\lambda = 1/[2(y_{\max} - y_{\min})^2]$.
- entropy loss: $\mathcal{Y} = [0; 1]$ and $\ell(y_1, y_2) = y_1 \log\big(\frac{y_1}{y_2}\big) + (1 - y_1) \log\big(\frac{1 - y_1}{1 - y_2}\big)$. Note that $\ell(0, 1) = \ell(1, 0) = +\infty$. Then we have $a = 1/2$ and may take $\lambda = 1$.
- exponential (or AdaBoost) loss: $\mathcal{Y} = [-y_{\max}; y_{\max}]$ and $\ell(y_1, y_2) = e^{-y_1 y_2}$. Then we have $a = 0$ and may take $\lambda = e^{-y_{\max}^2}$.
- logit loss: $\mathcal{Y} = [-y_{\max}; y_{\max}]$ and $\ell(y_1, y_2) = \log(1 + e^{-y_1 y_2})$. Then we have $a = 0$ and may take $\lambda = e^{-y_{\max}^2}$.

**Progressive indirect mixture rule.**   Let $\mathcal{G}$ be a finite reference set of prediction functions. Under the previous assumptions, the only known algorithms satisfying (1) are the progressive indirect mixture rules defined below.

For any $i \in \{0, \ldots, n\}$, the *cumulative loss* suffered by the prediction function $g$ on the first $i$ pairs of input-output is

$$\Sigma_i(g) \triangleq \sum_{j=1}^i \ell[Y_j, g(X_j)],$$

where by convention we take $\Sigma_0 \equiv 0$. Let $\pi$ denote the uniform distribution on $\mathcal{G}$. We define the probability distribution $\hat{\pi}_i$ on $\mathcal{G}$ as

$$\hat{\pi}_i \propto e^{-\lambda \Sigma_i} \cdot \pi$$

equivalently for any $g \in \mathcal{G}$, $\hat{\pi}_i(g) = e^{-\lambda \Sigma_i(g)}/(\sum_{g' \in \mathcal{G}} e^{-\lambda \Sigma_i(g')})$. This distribution concentrates on functions having low cumulative loss up to time $i$. For any $i \in \{0, \ldots, n\}$, let $\hat{h}_i$ be a prediction function such that

$$\forall (x, y) \in \mathcal{Z} \qquad \ell[y, \hat{h}_i(x)] \leq -\tfrac{1}{\lambda} \log \mathbb{E}_{g \sim \hat{\pi}_i} e^{-\lambda \ell[y, g(x)]}. \tag{2}$$

The *progressive indirect mixture rule* produces the prediction function

$$\hat{g}_{\mathrm{pim}} = \tfrac{1}{n+1} \sum_{i=0}^n \hat{h}_i.$$

From the uniform exp-concavity assumption and Jensen's inequality, $\hat{h}_i$ does exist since one may take $\hat{h}_i = \mathbb{E}_{g \sim \hat{\pi}_i} g$. This particular choice leads to the *progressive mixture rule*, for which the predicted output for any $x \in \mathcal{X}$ is

$$\hat{g}_{\mathrm{pm}}(x) = \sum_{g \in \mathcal{G}} \left( \tfrac{1}{n+1} \sum_{i=0}^n \frac{e^{-\lambda \Sigma_i(g)}}{\sum_{g' \in \mathcal{G}} e^{-\lambda \Sigma_i(g')}} \right) g(x).$$

Consequently, any result that holds for any progressive indirect mixture rule in particular holds for the progressive mixture rule.

The idea of a progressive mean of estimators has been introduced by Barron ([2]) in the context of density estimation with Kullback-Leibler loss. The form $\hat{g}_{\mathrm{pm}}$ is due to Catoni ([3]). It was also independently proposed in [4]. The study of this procedure was made in density estimation and least square regression in [5, 6, 7, 8]. Results for general losses can be found in [9, 10]. Finally, the progressive indirect mixture rule is inspired by the work of Vovk, Haussler, Kivinen and Warmuth [11, 12, 13] on sequential prediction and was studied in the "batch" setting in [10]. Finally, in the upper bounds we state, e.g. Inequality (1), one should notice that there is no constant larger than 1 in front of $\min_{g \in \mathcal{G}} R(g)$, as opposed to some existing upper bounds (e.g. [14]). This work really studies the behaviour of the excess risk, that is the random variable $R(\hat{g}) - \min_{g \in \mathcal{G}} R(g)$.

The largest integer smaller or equal to the logarithm in base 2 of $x$ is denoted by $\lfloor \log_2 x \rfloor$.

## 3 Expectation convergence rate

The following theorem, whose proof is omitted, shows that the expectation convergence rate of any progressive indirect mixture rule is (i) at least $(\log |\mathcal{G}|)/n$ and (ii) cannot be uniformly improved, even when we consider only probability distributions on $\mathcal{Z}$ for which the output has almost surely two symmetrical values (e.g. $\{-1;+1\}$ classication with exponential or logit losses).

**Theorem 1** *Any progressive indirect mixture rule satisfies*

$$\mathbb{E} R(\hat{g}_{pim}) \leq \min_{g \in \mathcal{G}} R(g) + \tfrac{\log |\mathcal{G}|}{\lambda(n+1)}.$$

*Let $y_1 \in \mathcal{Y} - \{a\}$ and $d$ be a positive integer. There exists a set $\mathcal{G}$ of $d$ prediction functions such that: for any learning algorithm, there exists a probability distribution generating the data for which*

- *the output marginal is supported by $2a - y_1$ and $y_1$: $P(Y \in \{2a - y_1; y_1\}) = 1$,*

- $\mathbb{E} R(\hat{g}) \geq \min_{g \in \mathcal{G}} R(g) + e^{-1} \kappa \big(1 \wedge \tfrac{\lfloor \log_2 |\mathcal{G}| \rfloor}{n+1}\big)$, *with $\kappa \triangleq \sup_{y \in \mathcal{Y}} [\ell(y_1, a) - \ell(y_1, y)] > 0$.*

The second part of Theorem 1 has the same $(\log |\mathcal{G}|)/n$ rate as the lower bounds obtained in sequential prediction ([12]). From the link between sequential predictions and our "batch" setting with i.i.d. data (see e.g. [10, Lemma 3]), upper bounds for sequential prediction lead to upper bounds for i.i.d. data, and lower bounds for i.i.d. data leads to lower bounds for sequential prediction. The converse of this last assertion is not true, so that the second part of Theorem 1 is not a consequence of the lower bounds of [12].

The following theorem, whose proof is also omitted, shows that for appropriate set $\mathcal{G}$: (i) the empirical risk minimizer has a $\sqrt{(\log|\mathcal{G}|)/n}$ expectation convergence rate, and (ii) any empirical risk minimizer and any of its penalized variants are really poor algorithms in our learning task since their expectation convergence rate cannot be faster than $\sqrt{(\log|\mathcal{G}|)/n}$ (see [5, p.14] and [1] for results of the same spirit). This last point explains the interest we have in progressive mixture rules.

**Theorem 2** *If $B \triangleq \sup_{y,y',y'' \in \mathcal{Y}}[\ell(y,y') - \ell(y,y'')] < +\infty$, then any empirical risk minimizer, which produces a prediction function $\hat{g}_{erm}$ in $\operatorname{argmin}_{g \in \mathcal{G}} \Sigma_n$, satisfies:*

$$\mathbb{E}R(\hat{g}_{erm}) \leq \min_{g \in \mathcal{G}} R(g) + B\sqrt{\frac{2\log|\mathcal{G}|}{n}}.$$

*Let $y_1, \tilde{y}_1 \in \mathcal{Y} \cap ]a; +\infty[$ and $d$ be a positive integer. There exists a set $\mathcal{G}$ of $d$ prediction functions such that: for any learning algorithm producing a prediction function in $\mathcal{G}$ (e.g. $\hat{g}_{erm}$) there exists a probability distribution generating the data for which*

- *the output marginal is supported by $2a - y_1$ and $y_1$: $P(Y \in \{2a - y_1; y_1\}) = 1$,*

- *$\mathbb{E}R(\hat{g}) \geq \min_{g \in \mathcal{G}} R(g) + \frac{\delta}{8}\left(\sqrt{\frac{\lfloor\log_2|\mathcal{G}|\rfloor}{n}} \wedge 2\right)$, with $\delta \triangleq \ell(y_1, 2a - \tilde{y}_1) - \ell(y_1, \tilde{y}_1) > 0$.*

The lower bound of Theorem 2 also says that one should not use cross-validation. This holds for the loss functions considered in this work, and not for, e.g., the classification loss: $\ell(y, y') = \mathbf{1}_{y \neq y'}$.

# 4 Deviation convergence rate

The following theorem shows that the deviation convergence rate of any progressive indirect mixture rule is (i) at least $1/\sqrt{n}$ and (ii) cannot be uniformly improved, even when we consider only probability distributions on $\mathcal{Z}$ for which the output has almost surely two symmetrical values (e.g. $\{-1;+1\}$ classication with exponential or logit losses).

**Theorem 3** *If $B \triangleq \sup_{y,y',y'' \in \mathcal{Y}}[\ell(y,y') - \ell(y,y'')] < +\infty$, then any progressive indirect mixture rule satisfies: for any $\epsilon > 0$, with probability at least $1 - \epsilon$ w.r.t. the training set distribution, we have*

$$R(\hat{g}_{pim}) \leq \min_{g \in \mathcal{G}} R(g) + B\sqrt{\frac{2\log(2\epsilon^{-1})}{n+1}} + \frac{\log|\mathcal{G}|}{\lambda(n+1)}$$

*Let $y_1$ and $\tilde{y}_1$ in $\mathcal{Y} \cap ]a; +\infty[$ such that $\ell_{y_1}$ is twice continuously differentiable on $[a; \tilde{y}_1]$ and $\ell'_{y_1}(\tilde{y}_1) \leq 0$ and $\ell''_{y_1}(\tilde{y}_1) > 0$. Consider the prediction functions $g_1 \equiv \tilde{y}_1$ and $g_2 \equiv 2a - \tilde{y}_1$. For any training set size $n$ large enough, there exist $\epsilon > 0$ and a distribution generating the data such that*

- *the output marginal is supported by $y_1$ and $2a - y_1$*

- *with probability larger than $\epsilon$, we have*

$$R(\hat{g}_{pim}) - \min_{g \in \{g_1, g_2\}} R(g) \geq c\sqrt{\frac{\log(e\epsilon^{-1})}{n}}$$

*where $c$ is a positive constant depending only on the loss function, the symmetry parameter $a$ and the output values $y_1$ and $\tilde{y}_1$.*

**Proof 1** *See Section 5.*

This result is quite surprising since it gives an example of an algorithm which is optimal in terms of expectation convergence rate and for which the deviation convergence rate is (significantly) worse than the expectation convergence rate.

In fact, despite their popularity based on their unique expectation convergence rate, the progressive mixture rules are not good algorithms since a long argument essentially based on convexity shows that the following algorithm has both expectation and deviation convergence rate of order $1/n$. Let

$\hat{g}_{\text{erm}}$ be the minimizer of the empirical risk among functions in $\mathcal{G}$. Let $\tilde{g}$ be the minimizer of the empirical risk in the star $\hat{\mathcal{G}} = \cup_{g \in \mathcal{G}} [g; \hat{g}_{\text{erm}}]$. The algorithm producing $\tilde{g}$ satisfies for some $C > 0$, for any $\epsilon > 0$, with probability at least $1 - \epsilon$ w.r.t. the training set distribution, we have

$$R(\tilde{g}) \leq \min_{g \in \mathcal{G}} R(g) + C \frac{\log(\epsilon^{-1}|\mathcal{G}|)}{n}.$$

This algorithm has also the benefit of being parameter-free. On the contrary, in practice, one will have recourse to cross-validation to tune the parameter $\lambda$ of the progressive mixture rule.

To summarize, to predict as well as the best prediction function in a given set $\mathcal{G}$, one should not restrain the algorithm to produce its prediction function among the set $\mathcal{G}$. The progressive mixture rules satisfy this principle since they produce a prediction function in the convex hull of $\mathcal{G}$. This allows to achieve $(\log |\mathcal{G}|)/n$ convergence rates in expectation. The proof of the lower bound of Theorem 3 shows that the progressive mixtures overfit the data: the deviations of their excess risk are not PAC bounded by $C \log(\epsilon^{-1}|\mathcal{G}|)/n$ while an appropriate algorithm producing prediction functions on the edges of the convex hull achieves the $\log(\epsilon^{-1}|\mathcal{G}|)/n$ deviation convergence rate. Future work might look at whether one can transpose this algorithm to the sequential prediction setting, in which, up to now, the algorithms to predict as well as the best expert were dominated by algorithms producing a mixture expert inside the convex hull of the set of experts.

## 5 Proof of Theorem 3

### 5.1 Proof of the upper bound

Let $Z_{n+1} = (X_{n+1}, Y_{n+1})$ be an input-output pair independent from the training set $Z_1, \ldots, Z_n$ and with the same distribution $P$. From the convexity of $y' \mapsto \ell(y, y')$, we have

$$R(\hat{g}_{\text{pim}}) \leq \frac{1}{n+1} \sum_{i=0}^{n} R(\hat{h}_i). \tag{3}$$

Now from [15, Theorem 1] (see also [16, Proposition 1]), for any $\epsilon > 0$, with probability at least $1 - \epsilon$, we have

$$\frac{1}{n+1} \sum_{i=0}^{n} R(\hat{h}_i) \leq \frac{1}{n+1} \sum_{i=0}^{n} \ell\big(Y_{i+1}, \hat{h}(X_{i+1})\big) + B\sqrt{\frac{\log(\epsilon^{-1})}{2(n+1)}} \tag{4}$$

Using [12, Theorem 3.8] and the exp-concavity assumption, we have

$$\sum_{i=0}^{n} \ell\big(Y_{i+1}, \hat{h}(X_{i+1})\big) \leq \min_{g \in \mathcal{G}} \sum_{i=0}^{n} \ell\big(Y_{i+1}, g(X_{i+1})\big) + \frac{\log |\mathcal{G}|}{\lambda} \tag{5}$$

Let $\tilde{g} \in \operatorname{argmin}_{\mathcal{G}} R$. By Hoeffding's inequality, with probability at least $1 - \epsilon$, we have

$$\frac{1}{n+1} \sum_{i=0}^{n} \ell\big(Y_{i+1}, \tilde{g}(X_{i+1})\big) \leq R(\tilde{g}) + B\sqrt{\frac{\log(\epsilon^{-1})}{2(n+1)}} \tag{6}$$

Merging (3), (4), (5) and (6), with probability at least $1 - 2\epsilon$, we get

$$
\begin{aligned}
R(\hat{g}_{\text{pim}}) &\leq \frac{1}{n+1} \sum_{i=0}^{n} \ell\big(Y_{i+1}, \tilde{g}(X_{i+1})\big) + \frac{\log |\mathcal{G}|}{\lambda(n+1)} + B\sqrt{\frac{\log(\epsilon^{-1})}{2(n+1)}} \\
&\leq R(\tilde{g}) + B\sqrt{\frac{2\log(\epsilon^{-1})}{n+1}} + \frac{\log |\mathcal{G}|}{\lambda(n+1)}.
\end{aligned}
$$

### 5.2 Sketch of the proof of the lower bound

We cannot use standard tools like Assouad's argument (see e.g. [17, Theorem 14.6]) because if it were possible, it would mean that the lower bound would hold for any algorithm and in particular for $\tilde{g}$, and this is false. To prove that any progressive indirect mixture rule have no fast exponential deviation inequalities, we will show that on some event with not too small probability, for most of the $i$ in $\{0, \ldots, n\}$, $\pi_{-\lambda \Sigma_i}$ concentrates on the wrong function.

The proof is organized as follows. First we define the probability distribution for which we will prove that the progressive indirect mixture rules cannot have fast deviation convergence rates. Then we define the event on which the progressive indirect mixture rules do not perform well. We lower bound the probability of this excursion event. Finally we conclude by lower bounding $R(\hat{g}_{\text{pim}})$ on the excursion event.

Before starting the proof, note that from the "well behaved at center" and exp-concavity assumptions, for any $y \in \mathcal{Y} \cap ]a; +\infty[$, on a neighborhood of $a$, we have: $\ell_y'' \geq \lambda(\ell_y')^2$ and since $\ell_y'(a) < 0$, $y_1$ and $\tilde{y}_1$ exist. Due to limited space, some technical computations have been removed.

### 5.2.1 Probability distribution generating the data and first consequences.

Let $\gamma \in ]0;1]$ be a parameter to be tuned later. We consider a distribution generating the data such that the output distribution satisfies for any $x \in \mathcal{X}$

$$P(Y = y_1 | X = x) = (1 + \gamma)/2 = 1 - P(Y = y_2 | X = x),$$

where $y_2 = 2a - y_1$. Let $\tilde{y}_2 = 2a - \tilde{y}_1$. From the symmetry and admissibility assumptions, we have $\ell(y_2, \tilde{y}_2) = \ell(y_1, \tilde{y}_1) < \ell(y_1, \tilde{y}_2) = \ell(y_2, \tilde{y}_1)$. Introduce

$$\delta \triangleq \ell(y_1, \tilde{y}_2) - \ell(y_1, \tilde{y}_1) > 0. \tag{7}$$

We have

$$R(g_2) - R(g_1) = \tfrac{1+\gamma}{2}[\ell(y_1, \tilde{y}_2) - \ell(y_1, \tilde{y}_1)] + \tfrac{1-\gamma}{2}[\ell(y_2, \tilde{y}_2) - \ell(y_2, \tilde{y}_1)] = \gamma\delta. \tag{8}$$

Therefore $g_1$ is the best prediction function in $\{g_1, g_2\}$ for the distribution we have chosen. Introduce $W_j \triangleq \mathbf{1}_{Y_j = y_1} - \mathbf{1}_{Y_j = y_2}$ and $S_i \triangleq \sum_{j=1}^{i} W_j$. For any $i \in \{1, \dots, n\}$, we have

$$\Sigma_i(g_2) - \Sigma_i(g_1) = \sum_{j=1}^{i}[\ell(Y_j, \tilde{y}_2) - \ell(Y_j, \tilde{y}_1)] = \sum_{j=1}^{i} W_j \delta = \delta\, S_i$$

The weight given by the Gibbs distribution $\pi_{-\lambda\Sigma_i}$ to the function $g_1$ is

$$\pi_{-\lambda\Sigma_i}(g_1) = \frac{e^{-\lambda\Sigma_i(g_1)}}{e^{-\lambda\Sigma_i(g_1)} + e^{-\lambda\Sigma_i(g_2)}} = \frac{1}{1 + e^{\lambda[\Sigma_i(g_1) - \Sigma_i(g_2)]}} = \frac{1}{1 + e^{-\lambda\delta S_i}}. \tag{9}$$

### 5.2.2 An excursion event on which the progressive indirect mixture rules will not perform well.

Equality (9) leads us to consider the event:

$$E_\tau = \big\{ \forall i \in \{\tau, \dots, n\}, \ S_i \leq -\tau \big\},$$

with $\tau$ the smallest integer larger than $(\log n)/(\lambda\delta)$ such that $n - \tau$ is even (for convenience). We have

$$\tfrac{\log n}{\lambda\delta} \leq \tau \leq \tfrac{\log n}{\lambda\delta} + 2. \tag{10}$$

The event $E_\tau$ can be seen as an excursion event of the random walk defined through the random variables $W_j = \mathbf{1}_{Y_j = y_1} - \mathbf{1}_{Y_j = y_2}, j \in \{1, \dots, n\}$, which are equal to $+1$ with probability $(1+\gamma)/2$ and $-1$ with probability $(1 - \gamma)/2$.

From (9), on the event $E_\tau$, for any $i \in \{\tau, \dots, n\}$, we have

$$\pi_{-\lambda\Sigma_i}(g_1) \leq \tfrac{1}{n+1}. \tag{11}$$

This means that $\pi_{-\lambda\Sigma_i}$ concentrates on the wrong function, i.e. the function $g_2$ having larger risk (see (8)).

### 5.2.3 Lower bound of the probability of the excursion event.

This requires to look at the probability that a slightly shifted random walk in the integer space has a very long excursion above a certain threshold. To lower bound this probability, we will first look at the non-shifted random walk. Then we will see that for small enough shift parameter, probabilities of shifted random walk events are close to the ones associated to the non-shifted random walk.

Let $N$ be a positive integer. Let $\sigma_1, \dots, \sigma_N$ be $N$ independent Rademacher variables: $\mathbb{P}(\sigma_i = +1) = \mathbb{P}(\sigma_i = -1) = 1/2$. Let $s_i \triangleq \sum_{j=1}^{i} \sigma_i$ be the sum of the first $i$ Rademacher variables. We start with the following lemma for sums of Rademacher variables (proof omitted).

**Lemma 1** *Let $m$ and $t$ be positive integers. We have*

$$\mathbb{P}\big(\max_{1 \leq k \leq N} s_k \geq t; s_N \neq t; \big|s_N - t\big| \leq m\big) = 2\mathbb{P}\big(t < s_N \leq t + m\big) \tag{12}$$

Let $\sigma'_1, \dots, \sigma'_N$ be $N$ independent shifted Rademacher variables to the extent that $\mathbb{P}(\sigma'_i = +1) = (1 + \gamma)/2 = 1 - \mathbb{P}(\sigma'_i = -1)$. These random variables satisfy the following key lemma (proof omitted)

**Lemma 2** *For any set* $A \subset \big\{(\epsilon_1, \ldots, \epsilon_N) \in \{-1,1\}^n : \big|\sum_{i=1}^N \epsilon_i\big| \le M\big\}$ *where $M$ is a positive integer, we have*

$$\mathbb{P}\big\{(\sigma'_1, \ldots, \sigma'_N) \in A\big\} \ge \Big(\tfrac{1-\gamma}{1+\gamma}\Big)^{M/2}\big(1-\gamma^2\big)^{N/2}\mathbb{P}\big\{(\sigma_1, \ldots, \sigma_N) \in A\big\} \tag{13}$$

We may now lower bound the probability of the excursion event $E_\tau$. Let $M$ be an integer larger than $\tau$. We still use $W_j \triangleq \mathbf{1}_{Y_j=y_1} - \mathbf{1}_{Y_j=y_2}$ for $j \in \{1, \ldots, n\}$. By using Lemma 2 with $N = n - 2\tau$, we obtain

$$
\begin{aligned}
\mathbb{P}(E_\tau) &\ge \mathbb{P}\big(W_1 = -1, \ldots, W_{2\tau} = -1; \forall 2\tau < i \le n, \textstyle\sum_{j=2\tau+1}^i W_j \le \tau\big) \\
&= \big(\tfrac{1-\gamma}{2}\big)^{2\tau}\mathbb{P}\big(\forall i \in \{1, \ldots, N\} \quad \textstyle\sum_{j=1}^i \sigma'_j \le \tau\big) \\
&\ge \big(\tfrac{1-\gamma}{2}\big)^{2\tau}\big(\tfrac{1-\gamma}{1+\gamma}\big)^{M/2}\big(1-\gamma^2\big)^{\frac{N}{2}}\mathbb{P}\big(|s_N| \le M; \forall i \in \{1, \ldots, N\} \quad s_i \le \tau\big)
\end{aligned}
$$

By using Lemma 1, since $\tau \le M$, the r.h.s. probability can be lower bounded, and after some computations, we obtain

$$\mathbb{P}(E_\tau) \ge \tau\big(\tfrac{1-\gamma}{2}\big)^{2\tau}\big(\tfrac{1-\gamma}{1+\gamma}\big)^{M/2}\big(1-\gamma^2\big)^{\frac{N}{2}}\big[\mathbb{P}(s_N = \tau) - \mathbb{P}(s_N = M)\big] \tag{14}$$

where we recall that $\tau$ have the order of $\log n$, $N = n - 2\tau$ has the order of $n$ and that $\gamma > 0$ and $M \ge \tau$ have to be appropriately chosen.

To control the probabilities of the r.h.s., we use Stirling's formula

$$n^n e^{-n}\sqrt{2\pi n}\, e^{1/(12n+1)} < n! < n^n e^{-n}\sqrt{2\pi n}\, e^{1/(12n)}, \tag{15}$$

and get for any $s \in [0; N]$ such that $N - s$ even,

$$\mathbb{P}(s_N = s) \ge \sqrt{\tfrac{2}{\pi N}}\Big(1 - \tfrac{s^2}{N^2}\Big)^{-\frac{N}{2}}\Big(\tfrac{1-\frac{s}{N}}{1+\frac{s}{N}}\Big)^{\frac{s}{2}}e^{-\frac{1}{6(N+s)} - \frac{1}{6(N-s)}} \tag{16}$$

and similarly

$$\mathbb{P}(s_N = s) \le \sqrt{\tfrac{2}{\pi N}}\Big(1 - \tfrac{s^2}{N^2}\Big)^{-\frac{N}{2}}\Big(\tfrac{1-\frac{s}{N}}{1+\frac{s}{N}}\Big)^{\frac{s}{2}}e^{\frac{1}{12N+1}}. \tag{17}$$

These computations and (14) leads us to take $M$ as the smallest integer larger than $\sqrt{n}$ such that $n - M$ is even. Indeed, from (10), (16) and (17), we obtain $\lim_{n \to +\infty}\sqrt{n}[\mathbb{P}(s_N = \tau) - \mathbb{P}(s_N = M)] = c$, where $c = \sqrt{2/\pi}\big(1 - e^{-1/2}\big) > 0$. Therefore for $n$ large enough we have

$$\mathbb{P}(E_\tau) \ge \tfrac{c\tau}{2\sqrt{n}}\big(\tfrac{1-\gamma}{2}\big)^{2\tau}\big(\tfrac{1-\gamma}{1+\gamma}\big)^{M/2}\big(1-\gamma^2\big)^{\frac{N}{2}} \tag{18}$$

The last two terms of the r.h.s. of (18) leads us to take $\gamma$ of order $1/\sqrt{n}$ up to possibly a logarithmic term. We obtain the following lower bound on the excursion probability

**Lemma 3** *If $\gamma = \sqrt{C_0(\log n)/n}$ with $C_0$ a positive constant, then for any large enough $n$,*

$$\mathbb{P}(E_\tau) \ge \tfrac{1}{n^{C_0}}.$$

#### 5.2.4 Behavior of the progressive indirect mixture rule on the excursion event.

From now on, we work on the event $E_\tau$. We have $\hat{g}_{\mathrm{pim}} = (\sum_{i=0}^n \hat{h}_i)/(n+1)$. We still use $\delta \triangleq \ell(y_1, \tilde{y}_2) - \ell(y_1, \tilde{y}_1) = \ell(y_2, \tilde{y}_1) - \ell(y_2, \tilde{y}_2)$. On the event $E_\tau$, for any $x \in \mathcal{X}$ and any $i \in \{\tau, \ldots, n\}$, by definition of $\hat{h}_i$, we have

$$
\begin{aligned}
\ell[y_2, \hat{h}_i(x)] - \ell(y_2, \tilde{y}_2) &\le -\tfrac{1}{\lambda}\log \mathbb{E}_{g \sim \pi_{-\lambda\Sigma_i}} e^{-\lambda\{\ell[y_2, g(x)] - \ell(y_2, \tilde{y}_2)\}} \\
&= -\tfrac{1}{\lambda}\log\big\{e^{-\lambda\delta} + (1 - e^{-\lambda\delta})\pi_{-\lambda\Sigma_i}(g_2)\big\} \\
&\le -\tfrac{1}{\lambda}\log\big\{1 - (1 - e^{-\lambda\delta})\tfrac{1}{n+1}\big\}
\end{aligned}
$$

In particular, for any $n$ large enough, we have $\ell[y_2, \hat{h}_i(x)] - \ell(y_2, \tilde{y}_2) \le Cn^{-1}$, with $C > 0$ *independent from $\gamma$*. From the convexity of the function $y \mapsto \ell(y_2, y)$ and by Jensen's inequality, we obtain

$$\ell[y_2, \hat{g}_{\mathrm{pim}}(x)] - \ell(y_2, \tilde{y}_2) \le \tfrac{1}{n+1}\textstyle\sum_{i=0}^n \ell[y_2, \hat{h}_i(x)] - \ell(y_2, \tilde{y}_2) \le \tfrac{\tau\delta}{n+1} + Cn^{-1} < C_1\tfrac{\log n}{n}$$

for some constant $C_1 > 0$ *independent from* $\gamma$. Let us now prove that for $n$ large enough, we have

$$\tilde{y_2} \le \hat{g}_{\text{pim}}(x) \le \tilde{y_2} + C\sqrt{\tfrac{\log n}{n}} \le \tilde{y_1}, \tag{19}$$

with $C > 0$ *independent from* $\gamma$.

From (19), we obtain

$$
\begin{aligned}
R(\hat{g}_{\text{pim}}) - R(g_1) &= \tfrac{1+\gamma}{2}\big[\ell(y_1, \hat{g}_{\text{pim}}) - \ell(y_1, \tilde{y_1})\big] + \tfrac{1-\gamma}{2}\big[\ell(y_2, \hat{g}_{\text{pim}}) - \ell(y_2, \tilde{y_1})\big]\\
&= \tfrac{1+\gamma}{2}\big[\ell_{y_1}(\hat{g}_{\text{pim}}) - \ell_{y_1}(\tilde{y_1})\big] + \tfrac{1-\gamma}{2}\big[\ell_{y_1}(2a - \hat{g}_{\text{pim}}) - \ell_{y_1}(\tilde{y_2})\big]\\
&= \tfrac{1+\gamma}{2}\big[\delta + \ell_{y_1}(\hat{g}_{\text{pim}}) - \ell_{y_1}(\tilde{y_2})\big] + \tfrac{1-\gamma}{2}\big[-\delta + \ell_{y_1}(2a - \hat{g}_{\text{pim}}) - \ell_{y_1}(\tilde{y_1})\big]\\
&\ge \gamma\delta - (\hat{g}_{\text{pim}} - \tilde{y_2})|\ell'_{y_1}(\tilde{y_2})|\\
&\ge \gamma\delta - C_2\sqrt{\tfrac{\log n}{n}},
\end{aligned}
\tag{20}
$$

with $C_2$ *independent from* $\gamma$. We may take $\gamma = \frac{2C_2}{\delta}\sqrt{(\log n)/n}$ and obtain: for $n$ large enough, on the event $E_\tau$, we have $R(\hat{g}_{\text{pim}}) - R(g_1) \ge C\sqrt{\log n/n}$. From Lemma 3, this inequality holds with probability at least $1/n^{C_4}$ for some $C_4 > 0$. To conclude, for any $n$ large enough, there exists $\epsilon > 0$ s.t. with probability at least $\epsilon$, $R(\hat{g}_{\text{pim}}) - R(g_1) \ge c\sqrt{\frac{\log(e\epsilon^{-1})}{n}}$. where $c$ is a positive constant depending only on the loss function, the symmetry parameter $a$ and the output values $y_1$ and $\tilde{y_1}$.

## Footnotes

[1] Indeed, if $\xi$ denotes the function $e^{-\lambda \ell_y}$, from Jensen's inequality, for any probability distribution, $\mathbb{E}\ell_y(Y) = \mathbb{E}\big(-\frac{1}{\lambda} \log \xi(Y)\big) \geq -\frac{1}{\lambda} \log \mathbb{E}\xi(Y) \geq -\frac{1}{\lambda} \log \xi(\mathbb{E}Y) = \ell_y(\mathbb{E}Y)$.

## References

[1] G. Lecué. Suboptimality of penalized empirical risk minimization in classification. In *Proceedings of the 20th annual conference on Computational Learning Theory*, 2007.

[2] A. Barron. Are bayes rules consistent in information? In T.M. Cover and B. Gopinath, editors, *Open Problems in Communication and Computation*, pages 85–91. Springer, 1987.

[3] O. Catoni. A mixture approach to universal model selection. preprint LMENS 97-30, Available from `http://www.dma.ens.fr/edition/preprints/Index.97.html`, 1997.

[4] A. Barron and Y. Yang. Information-theoretic determination of minimax rates of convergence. *Ann. Stat.*, 27(5):1564–1599, 1999.

[5] O. Catoni. Universal aggregation rules with exact bias bound. Preprint n.510, `http://www.proba.jussieu.fr/mathdoc/preprints/index.html\#1999`, 1999.

[6] G. Blanchard. The progressive mixture estimator for regression trees. *Ann. Inst. Henri Poincaré, Probab. Stat.*, 35(6):793–820, 1999.

[7] Y. Yang. Combining different procedures for adaptive regression. *Journal of multivariate analysis*, 74:135–161, 2000.

[8] F. Bunea and A. Nobel. Sequential procedures for aggregating arbitrary estimators of a conditional mean, 2005. Technical report.

[9] A. Juditsky, P. Rigollet, and A.B. Tsybakov. Learning by mirror averaging. Preprint n.1034, Laboratoire de Probabilités et Modèles Aléatoires, Universités Paris 6 and Paris 7, 2005.

[10] J.-Y. Audibert. A randomized online learning algorithm for better variance control. In *Proceedings of the 19th annual conference on Computational Learning Theory*, pages 392–407, 2006.

[11] V.G. Vovk. Aggregating strategies. In *Proceedings of the 3rd annual workshop on Computational Learning Theory*, pages 371–386, 1990.

[12] D. Haussler, J. Kivinen, and M. K. Warmuth. Sequential prediction of individual sequences under general loss functions. *IEEE Trans. on Information Theory*, 44(5):1906–1925, 1998.

[13] V.G. Vovk. A game of prediction with expert advice. *Journal of Computer and System Sciences*, pages 153–173, 1998.

[14] M. Wegkamp. Model selection in nonparametric regression. *Ann. Stat.*, 31(1):252–273, 2003.

[15] T. Zhang. Data dependent concentration bounds for sequential prediction algorithms. In *Proceedings of the 18th annual conference on Computational Learning Theory*, pages 173–187, 2005.

[16] N. Cesa-Bianchi, A. Conconi, and C. Gentile. On the generalization ability of on-line learning algorithms. *IEEE Transactions on Information Theory*, 50(9):2050–2057, 2004.

[17] L. Devroye, L. Györfi, and G. Lugosi. *A Probabilistic Theory of Pattern Recognition*. Springer-Verlag, 1996.

